# Tiled convolutional neural networks

**Quoc V. Le, Jiquan Ngiam, Zhenghao Chen, Daniel Chia, Pang Wei Koh, Andrew Y. Ng**
Computer Science Department, Stanford University
{quocle,jngiam,zhenghao,danchia,pangwei,ang}@cs.stanford.edu

## Abstract

Convolutional neural networks (CNNs) have been successfully applied to many tasks such as digit and object recognition. Using convolutional (tied) weights significantly reduces the number of parameters that have to be learned, and also allows translational invariance to be hard-coded into the architecture. In this paper, we consider the problem of learning invariances, rather than relying on hard-coding. We propose *tiled* convolution neural networks (Tiled CNNs), which use a regular "tiled" pattern of tied weights that does not require that adjacent hidden units share identical weights, but instead requires only that hidden units $k$ steps away from each other to have tied weights. By pooling over neighboring units, this architecture is able to learn complex invariances (such as scale and rotational invariance) beyond translational invariance. Further, it also enjoys much of CNNs' advantage of having a relatively small number of learned parameters (such as ease of learning and greater scalability). We provide an efficient learning algorithm for Tiled CNNs based on Topographic ICA, and show that learning complex invariant features allows us to achieve highly competitive results for both the NORB and CIFAR-10 datasets.

## 1 Introduction

Convolutional neural networks (CNNs) [1] have been successfully applied to many recognition tasks. These tasks include digit recognition (MNIST dataset [2]), object recognition (NORB dataset [3]), and natural language processing [4]. CNNs take translated versions of the same basis function, and "pool" over them to build translational invariant features. By sharing the same basis function across different image locations (weight-tying), CNNs have significantly fewer learnable parameters which makes it possible to train them with fewer examples than if entirely different basis functions were learned at different locations (untied weights). Furthermore, CNNs naturally enjoy translational invariance, since this is hard-coded into the network architecture. However, one disadvantage of this hard-coding approach is that the pooling architecture captures *only* translational invariance; the network does not, for example, pool across units that are rotations of each other or capture more complex invariances, such as out-of-plane rotations.

Is it better to hard-code translational invariance – since this is a useful form of prior knowledge – or let the network learn its own invariances from unlabeled data? In this paper, we show that the latter is superior and describe an algorithm that can do so, outperforming convolutional methods. In particular, we present *tiled* convolutional networks (Tiled CNNs), which use a novel weight-tying scheme ("tiling") that simultaneously enjoys the benefit of significantly reducing the number of learnable parameters while giving the algorithm flexibility to learn other invariances. Our method is based on only constraining weights/basis functions $k$ steps away from each other to be equal (with the special case of $k = 1$ corresponding to convolutional networks).

In order to learn these invariances from unlabeled data, we employ unsupervised pretraining, which has been shown to help performance [5, 6, 7]. In particular, we use a modification of Topographic ICA (TICA) [8], which learns to organize features in a topographical map by pooling together groups

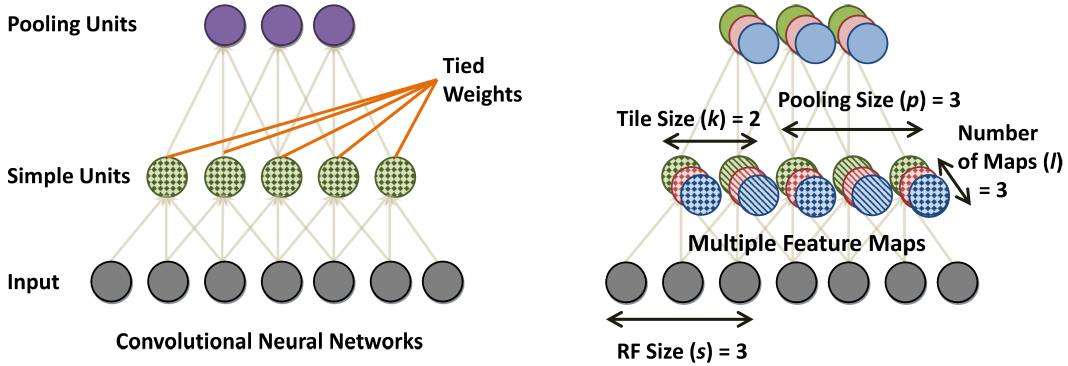

Figure 1: Left: Convolutional Neural Networks with local receptive fields and tied weights. Right: Partially untied local receptive field networks – Tiled CNNs. Units with the same color belong to the same map; within each map, units with the same fill texture have tied weights. (Network diagrams in the paper are shown in 1D for clarity.)

of related features. By pooling together local groups of features, it produces representations that are robust to local transformations [9]. We show in this paper how TICA can be efficiently used to pretrain Tiled CNNs through the use of local orthogonality.

The resulting Tiled CNNs pretrained with TICA are indeed able to learn invariant representations, with pooling units that are robust to both scaling and rotation. We find that this improves classification performance, enabling Tiled CNNs to be competitive with previously published results on the NORB [3] and CIFAR-10 [10] datasets.

## 2 Tiled CNNs

CNNs [1, 11] are based on two key concepts: local receptive fields, and weight-tying. Using local receptive fields means that each unit in the network only "looks" at a small, localized region of the input image. This is more computationally efficient than having full receptive fields, and allows CNNs to scale up well. Weight-tying additionally enforces that each first-layer (simple) unit shares the same weights (see Figure 1-Left). This reduces the number of learnable parameters, and (by pooling over neighboring units) further hard-codes translational invariance into the model.

Even though weight-tying allows one to hard-code translational invariance, it also prevents the pooling units from capturing more complex invariances, such as scale and rotation invariance. This is because the second layer units are constrained to pool over translations of identical bases. In this paper, rather than tying *all* of the weights in the network together, we instead develop a method that leaves nearby bases untied, but far-apart bases tied. This lets second-layer units pool over simple units that have different basis functions, and hence learn a more complex range of invariances.

We call this local untying of weights "tiling." Tiled CNNs are parametrized by a tile size $k$: we constrain only units that are $k$ steps away from each other to be tied. By varying $k$, we obtain a spectrum of models which trade off between being able to learn complex invariances, and having few learnable parameters. At one end of the spectrum we have traditional CNNs ($k = 1$), and at the other, we have fully untied simple units.

Next, we will allow our model to use multiple "maps," so as to learn highly overcomplete representations. A map is a set of pooling units and simple units that collectively cover the entire image (see Figure 1-Right). When varying the tiling size, we change the degree of weight tying within each map; for example, if $k = 1$, the simple units within each map will have the same weights. In our model, simple units in different maps are never tied. By having units in different maps learn different features, our model can learn a rich and diverse set of features. Tiled CNNs with multiple maps enjoy the twin benefits of (i) being able to represent complex invariances, by pooling over (partially) untied weights, and (ii) having a relatively small number of learnable parameters.

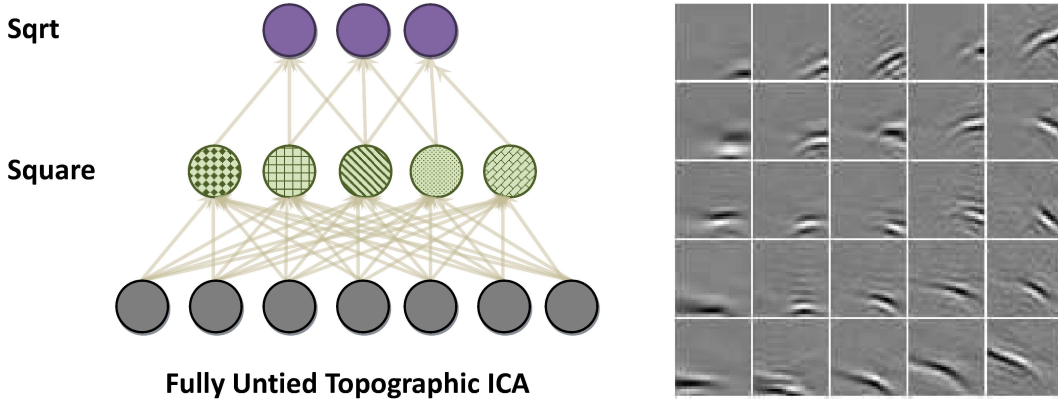

**Sqrt**

**Square**

**Fully Untied Topographic ICA**

Figure 2: Left: TICA network architecture. Right: TICA first layer filters (2D topography, 25 rows of $W$).

Unfortunately, existing methods for pretraining CNNs [11, 12] are not suitable for untied weights; for example, the CDBN algorithm [11] breaks down without the weight-tying constraints. In the following sections, we discuss a pretraining method for Tiled CNNs based on the TICA algorithm.

## 3   Unsupervised feature learning via TICA

TICA is an unsupervised learning algorithm that learns features from unlabeled image patches. A TICA network [9] can be described as a two-layered network (Figure 2-Left), with square and square-root nonlinearities in the first and second layers respectively. The weights $W$ in the first layer are learned, while the weights $V$ in the second layer are fixed and hard-coded to represent the neighborhood/topographical structure of the neurons in the first layer. Specifically, each second layer hidden unit $p_i$ pools over a small neighborhood of adjacent first layer units $h_i$. We call the $h_i$ and $p_i$ simple and pooling units, respectively.

More precisely, given an input pattern $x^{(t)}$, the activation of each second layer unit is $p_i(x^{(t)}; W, V) = \sqrt{\sum_{k=1}^{m} V_{ik}(\sum_{j=1}^{n} W_{kj} x_j^{(t)})^2}$. TICA learns the parameters $W$ through finding sparse feature representations in the second layer, by solving:

$$\underset{W}{\text{minimize}} \quad \sum_{t=1}^{T} \sum_{i=1}^{m} p_i(x^{(t)}; W, V), \ \text{subject to} \quad WW^T = \mathbf{I} \tag{1}$$

where the input patterns $\{x^{(t)}\}_{t=1}^{T}$ are whitened.[1] Here, $W \in \mathbb{R}^{m \times n}$ and $V \in \mathbb{R}^{m \times m}$, where $n$ is the size of the input and $m$ is the number of hidden units in a layer. $V$ is a fixed matrix ($V_{ij} = 1$ or $0$) that encodes the 2D topography of the hidden units $h_i$. Specifically, the $h_i$ units lie on a 2D grid, with each $p_i$ connected to a contiguous 3x3 (or other size) block of $h_i$ units.[2] The case of each $p_i$ being connected to exactly one $h_i$ corresponds to standard ICA. The orthogonality constraint $WW^T = \mathbf{I}$ provides competitiveness and ensures that the learned features are diverse.

One important property of TICA is that it can learn invariances even when trained only on unlabeled data, as demonstrated in [8, 9]. This is due both to the pooling architecture, which gives rise to pooling units that are robust to local transformations of their inputs, and the learning algorithm, which promotes selectivity by optimizing for sparsity. This combination of robustness and selectivity is central to feature invariance, which is in turn essential for recognition tasks [13].

If we choose square and square-root activations for the simple and pooling units in the Tiled CNN, we can view the Tiled CNN as a special case of a TICA network, with the topography of the pooling units specifying the matrix $V$.[3] Crucially, Tiled CNNs incorporate local receptive fields, which play an important role in speeding up TICA. We discuss this next.

## 4 Local receptive fields in TICA

Tiled CNNs typically perform much better at object recognition when the learned representation consists of multiple feature maps (Figure 1-Right). This corresponds to training TICA with an *overcomplete* representation ($m > n$). When learning overcomplete representations [14], the orthogonality constraint cannot be satisfied exactly, and we instead try to satisfy an approximate orthogonality constraint [15]. Unfortunately, these approximate orthogonality constraints are computationally expensive and have hyperparameters which need to be extensively tuned. Much of this tuning can be avoided by using score matching [16], but this is computationally even more expensive, and while orthogonalization can be avoided altogether with topographic sparse coding, those models are also expensive as they require further work either for inference at prediction time [9, 14] or for learning a decoder unit at training time [17].

We can avoid approximate orthogonalization by using local receptive fields, which are inherently built into Tiled CNNs. With these, the weight matrix $W$ for each simple unit is constrained to be 0 outside a small local region. This locality constraint automatically ensures that the weights of any two simple units with non-overlapping receptive fields are orthogonal, without the need for an explicit orthogonality constraint. Empirically, we find that orthogonalizing partially overlapping receptive fields is not necessary for learning distinct, informative features either.

However, orthogonalization is still needed to decorrelate units that occupy the same position in their respective maps, for they look at the same region on the image. Fortunately, this *local* orthogonalization is cheap: for example, if there are $l$ maps and if each receptive field is restricted to look at an input patch that contains $s$ pixels, we would only need to orthogonalize the rows of a $l$-by-$s$ matrix to ensure that the $l$ features over these $s$ pixels are orthogonal. Specifically, so long as $l \leq s$, we can demand that these $l$ units that share an input patch be orthogonal. Using this method, we can learn networks that are overcomplete by a factor of about $s$ (i.e., by learning $l = s$ maps), while having to orthogonalize only matrices that are $l$-by-$s$. This is significantly lower in cost than standard TICA. For $l$ maps, our computational cost is $O(ls^2n)$, compared to standard TICA's $O(l^2n^3)$.

In general, we will have $l \times k \times s$ learnable parameters for an input of size $n$. We note that setting $k$ to its maximum value of $n - s + 1$ gives exactly the untied local TICA model outlined in the previous section.[4]

## 5 Pretraining Tiled CNNs with local TICA

---
**Algorithm 1** Unsupervised pretraining of Tiled CNNs with TICA (line search)

---
**Input:** $\{x^{(t)}\}_{t=1}^{T}, W, V, k, s$        // $k$ is the tile size, $s$ is the receptive field size
**Output:** $W$
  **repeat**

$f^{old} \leftarrow \sum_{t=1}^{T}\sum_{i=1}^{m}\sqrt{\sum_{k=1}^{m}V_{ik}\left(\sum_{j=1}^{n}W_{kj}x_j^{(t)}\right)^2}, \quad g \leftarrow \frac{\partial\left[\sum_{t=1}^{T}\sum_{i=1}^{m}\sqrt{\sum_{k=1}^{m}V_{ik}\left(\sum_{j=1}^{n}W_{kj}x_j^{(t)}\right)^2}\right]}{\partial W}$

$f^{new} \leftarrow +\infty, \quad \alpha \leftarrow 1$
    **while** $f^{new} \geq f^{old}$ **do**
      $W^{new} \leftarrow W - \alpha g$
      $W^{new} \leftarrow localize(W^{new}, s)$
      $W^{new} \leftarrow tie\_weights(W^{new}, k)$
      $W^{new} \leftarrow orthogonalize\_local\_RF(W^{new})$

      $f^{new} \leftarrow \sum_{t=1}^{T}\sum_{i=1}^{m}\sqrt{\sum_{k=1}^{m}V_{ik}\left(\sum_{j=1}^{n}W_{kj}^{new}x_j^{(t)}\right)^2}$

      $\alpha \leftarrow 0.5\alpha$
    **end while**
    $W \leftarrow W^{new}$
  **until** convergence

---

Our pretraining algorithm, which is based on gradient descent on the TICA objective function (1), is shown in Algorithm 1. The innermost loop is a simple implementation of backtracking linesearch.

---

fields tend to be very localized, even without any explicit locality constraint. For example, when trained on natural images, TICA's first layer weights usually resemble localized Gabor filters (Figure 2-Right).

    [4]For a 2D input image of size $nxn$ and local RF of size $sxs$, the maximum value of $k$ is $(n - s + 1)^2$.

In $orthogonalize\_local\_RF(W^{new})$, we only orthogonalize the weights that have completely overlapping receptive fields. In $tie\_weights$, we enforce weight-tying by averaging each set of tied weights.

The algorithm is trained by batch projected gradient descent and usually requires little tuning of optimization parameters. This is because TICA's tractable objective function allows us to monitor convergence easily. In contrast, other unsupervised feature learning algorithms such as RBMs [6] and autoencoders [18] require much more parameter tuning, especially during optimization.

## 6 Experiments

### 6.1 Speed-up

We first establish that the local receptive fields intrinsic to Tiled CNNs allows us to implement TICA learning for overcomplete representations in a much more efficient manner.

Figure 3 shows the relative speed-up of pretraining Tiled CNNs over standard TICA using approximate fixed-point orthogonalization ($W = \frac{3}{2}W - \frac{1}{2}WW^TW$)[15]. These experiments were run on 10000 images of size 32x32 or 50x50, with $s = 8$.

We note that the weights in this experiment were left fully untied, i.e., $k = n-s+1$. Hence, the speed-up observed here is not from an efficient convolutional implementation, but purely due to the local receptive fields. Overcoming this computational challenge is the key that allows Tiled CNNs to successfully use TICA to learn features from unlabeled data.[5]

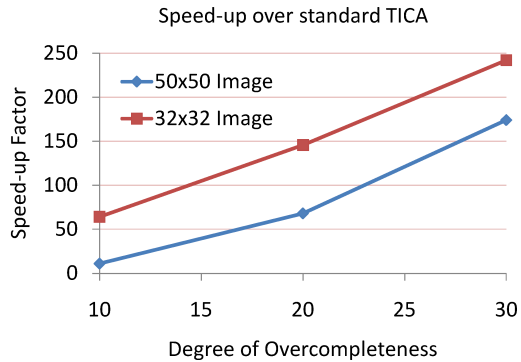

Figure 3: Speed-up of Tiled CNNs compared to standard TICA.

### 6.2 Classification on NORB

Next, we show that TICA pretraining for Tiled CNNs performs well on object recognition. We start with the *normalized-uniform* set for NORB, which consists of 24300 training examples and 24300 test examples drawn from 5 categories. In our case, each example is a preprocessed pair of 32x32 images.[6]

In our classification experiments, we fix the size of each local receptive field to 8x8, and set $V$ such that each pooling unit $p_i$ in the second layer pools over a block of 3x3 simple units in the first layer, without wraparound at the borders. The number of pooling units in each map is exactly the same as the number of simple units. We densely tile the input images with overlapping 8x8 local receptive fields, with a step size (or "stride") of 1. This gives us $25 \times 25 = 625$ simple units and 625 pooling units per map in our experiments on 32x32 images.

A summary of results is reported in Table 1.

### 6.2.1 Unsupervised pretraining

We first consider the case in which the features are learned purely from unsupervised data. In particular, we use the NORB training set itself (without the labels) as a source of unsupervised data

Table 1: Test set accuracy on NORB

| Algorithm | Accuracy |
|---|---|
| Tiled CNNs (with finetuning) (Section 6.2.2) | 96.1% |
| Tiled CNNs (without finetuning) (Section 6.2.1) | 94.5% |
| Standard TICA (10x overcomplete) | 89.6% |
| Convolutional Neural Networks [19], [12] | 94.1% , 94.4% |
| 3D Deep Belief Networks [19] | 93.5% |
| Support Vector Machines [20] | 88.4% |
| Deep Boltzmann Machines [21] | 92.8 % |

with which to learn the weights $W$ of the Tiled CNN. We call this initial phase the unsupervised pretraining phase.

After learning a feature representation from the unlabeled data, we train a linear classifier on the output of the Tiled CNN network (i.e., the activations of the pooling units) on the labeled training set. During this supervised training phase, only the weights of the linear classifier were learned, while the lower weights of the Tiled CNN model remained fixed.

We train a range of models to investigate the role of the tile size $k$ and the number of maps $l$.[7] The test set accuracy results of these models are shown in Figure 4-Left. Using a randomly sampled hold-out validation set of $2430$ examples ($10\%$) taken from the training set, we selected a convolutional model with 48 maps that achieved an accuracy of $94.5\%$ on the test set, indicating that Tiled CNNs learned purely on unsupervised data compare favorably to many state-of-the-art algorithms on NORB.

### 6.2.2 Supervised finetuning of W

Next, we study the effects of supervised finetuning [23] on the models produced by the unsupervised pretraining phase. Supervised finetuning takes place after unsupervised pretraining, but before the supervised training of the classifier.

Using softmax regression to calculate the gradients, we backpropagated the error signal from the output back to the learned features in order to update $W$, the weights of the simple units in the Tiled CNN model. During the finetuning step, the weights $W$ were adjusted without orthogonalization.

The results of supervised finetuning on our models are shown in Figure 4-Right. As above, we used a validation set comprising $10\%$ of the training data for model selection. Models with larger numbers of maps tended to overfit and hence performed poorly on the validation set. The best performing fine-tuned model on the validation set was the model with 16 maps and $k = 2$, which achieved a test-set accuracy of $96.1\%$. This substantially outperforms standard TICA, as well as the best published results on NORB to this date (see Table 1).

### 6.2.3 Limited training data

To test the ability of our pretrained features to generalize across rotations and lighting conditions given only a weak supervised signal, we limited the labeled training set to comprise only examples with a particular set of viewing angles and lighting conditions. Specifically, NORB contains images spanning 9 elevations, 18 azimuths and 6 lighting conditions, and we trained our linear classifier only on data with elevations $\{2, 4, 6\}$, azimuths $\{10, 18, 24\}$ and

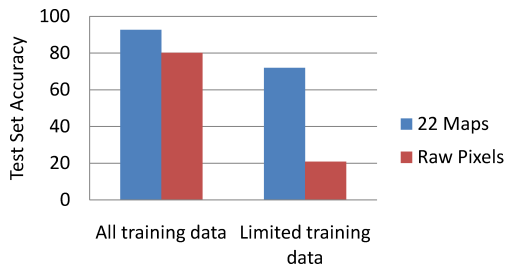

Figure 5: Test set accuracy on full and limited training sets

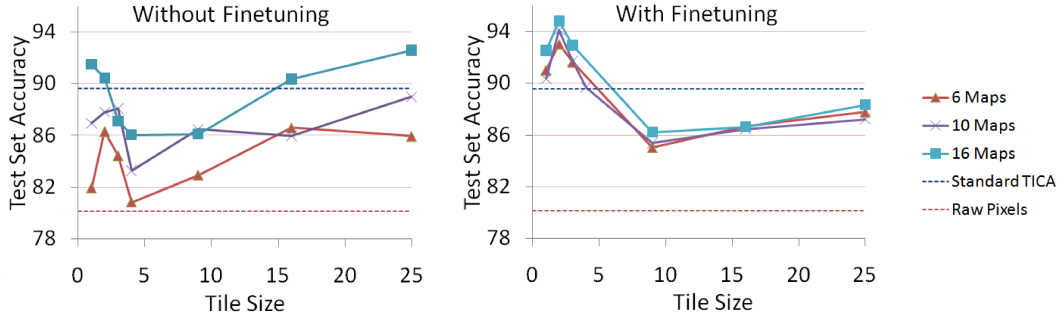

Figure 4: Left: NORB test set accuracy across various tile sizes and numbers of maps, without finetuning. Right: NORB test set accuracy, with finetuning.

lighting conditions $\{1, 3, 5\}$. Thus, for each object instance, the linear classifier sees only 27 training images, making for a total of 675 out of the possible 24300 training examples.

Using the pretrained network in Section 6.2.1, we trained a linear classifier on these 675 labeled examples. We obtained an accuracy of 72.2% on the full test set using the model with $k = 2$ and 22 maps. A smaller, approximately 2.5x overcomplete model with $k = 2$ and 4 maps obtained an accuracy of 64.9%. In stark contrast, raw pixel performance dropped sharply from 80.2% with a full supervised training set, to a near-chance level of 20.8% on this limited training set (Figure 5).

These results demonstrate that Tiled CNNs perform well even with limited labeled data. This is most likely because the partial weight-tying results in a relatively small number of learnable parameters, reducing the need for large amounts of labeled data.

## 6.3 Classification on CIFAR-10

Table 2: Test set accuracy on CIFAR-10

| Algorithm | Accuracy |
|---|---|
| Deep Tiled CNNs (s=4, with finetuning) (Section 6.3.2) | 73.1% |
| Tiled CNNs (s=8, without finetuning) (Section 6.3.1) | 66.1% |
| Standard TICA (10x, fixed-point orthogonalization) | 56.1% |
| Raw pixels [10] | 41.1% |
| RBM (one layer, 10000 units, finetuning) [10] | 64.8% |
| RBM (two layers, 10000 units, finetuning both layers) [10] | 60.3% |
| RBM (two layers, 10000 units, finetuning top layer) [10] | 62.2% |
| mcRBM (convolutional, trained on two million tiny images) [24] | 71.0% |
| Local Coordinate Coding (LCC) [25] | 72.3% |
| Improved Local Coordinate Coding (Improved LCC) [25] | 74.5% |

The CIFAR-10 dataset contains 50000 training images and 10000 test images drawn from 10 categories.[8] A summary of results for is reported in Table 2.

### 6.3.1 Unsupervised pretraining and supervised finetuning

As before, models were trained with tile size $k \in \{1, 2, 25\}$, and number of maps $l \in \{4, 10, 16, 22, 32\}$. The convolutional model ($k = 1$) was also trained with $l = 48$ maps. This 48-map convolutional model performed the best on our 10% hold-out validation set, and achieved a test set accuracy of $66.1\%$. We find that supervised finetuning of these models on CIFAR-10 causes overfitting, and generally reduces test-set accuracy; the top model on the validation set, with 32 maps and $k = 1$, only achieves $65.1\%$.

### 6.3.2 Deep Tiled CNNs

We additionally investigate the possibility of training a deep Tiled CNN in a greedy layer-wise fashion, similar to models such as DBNs [6] and stacked autoencoders [26, 18]. We constructed this network by stacking two Tiled CNNs, each with 10 maps and $k = 2$. The resulting four-layer network has the structure $W_1 \rightarrow V_1 \rightarrow W_2 \rightarrow V_2$, where the weights $W_1$ are local receptive fields of size 4x4, and $W_2$ is of size 3x3, i.e., each unit in the third layer "looks" at a 3x3 window of each of the 10 maps in the first layer. These parameters were chosen by an efficient architecture search [27] on the hold-out validation set. The number of maps in the third and fourth layer is also 10.

After finetuning, we found that the deep model outperformed all previous models on the validation set, and achieved a test set accuracy of $73.1\%$. This demonstrates the potential of deep Tiled CNNs to learn more complex representations.

### 6.4 Effects of optimizing the pooling units

When the tile size is 1 (i.e., a fully tied model), a naïve approach to learn the filter weights is to directly train the first layer filters using small patches (e.g., 8x8) randomly sampled from the dataset, with a method such as ICA. This method is computationally more attractive and probably easier to implement. Here, we investigate if such benefits come at the expense of classification accuracy.

We use ICA to learn the first layer weights on CIFAR-10 with 16 filters. These weights are then used in a Tiled CNN with a tile size of 1 and 16 maps. This method is compared to pretraining the model of the same architecture with TICA. For both methods, we do not use finetuning. Interestingly, classification on the test set show that the naïve approach results in significantly reduced classification accuracy: the naïve approach obtains $51.54\%$ on the test set, while pretraining with TICA achieves $58.66\%$. These results confirm that optimizing for sparsity of the pooling units results in better features than just naïvely approximating the first layer weights.

## 7 Discussion and Conclusion

Our results show that untying weights is beneficial for classification performance. Specifically, we find that selecting a tile size of $k = 2$ achieves the best results for both the NORB and CIFAR-10 datasets, even with deep networks. More importantly, untying weights allow the networks to learn more complex invariances from unlabeled data. By visualizing [28, 29] the range of optimal stimulus that activate each pooling unit in a Tiled CNN, we found units that were scale and rotationally invariant.[9] We note that a standard CNN is unlikely to be invariant to these transformations.

A natural choice of the tile size $k$ would be to set it to the size of the pooling region $p$, which in this case is 3. In this case, each pooling unit always combines simple units which are not tied. However, increasing the tile size leads to a higher degree of freedom in the models, making them susceptible to overfitting (learning unwanted non-stationary statistics of the dataset). Fortunately, the Tiled CNN only requires unlabeled data for training, which can be obtained cheaply. Our preliminary results on networks pretrained using 250000 unlabeled images from the Tiny images dataset [30] show that performance increases as $k$ goes from 1 to 3, flattening out at $k = 4$. This suggests that when there is sufficient data to avoid overfitting, setting $k = p$ can be a very good choice.

In this paper, we introduced Tiled CNNs as an extension of CNNs that support both unsupervised pretraining and weight tiling. The idea of tiling, or partial untying of filter weights, is a parametrization of a spectrum of models which includes both fully-convolutional and fully-untied weight schemes as natural special cases. Furthermore, the use of local receptive fields enable our models to scale up well, producing massively overcomplete representations that perform well on classification tasks. These principles allow Tiled CNNs to achieve competitive results on the NORB and CIFAR-10 object recognition datasets. Importantly, tiling is directly applicable and can potentially benefit a wide range of other feature learning models.

**Acknowledgements:** We thank Adam Coates, David Kamm, Andrew Maas, Andrew Saxe, Serena Yeung and Chenguang Zhu for insightful discussions. This work was supported by the DARPA Deep Learning program under contract number FA8650-10-C-7020.

## Footnotes

[1] Whitening means that they have been linearly transformed to have zero mean and identity covariance.

[2] For illustration, however, the figures in this paper depict $x_i$, $h_i$ and $p_i$ in 1D and show a 1D topography.

[3] The locality constraint, in addition to being biologically motivated by the receptive field organization patterns in V1, is also a natural approximation to the original TICA algorithm as the original learned receptive

[5]All algorithms are implemented in MATLAB, and executed on a computer with 3.0GHz CPU, 9Gb RAM. While orthogonalization alone is $10^4$ times faster in Tiled CNNs, other computations such as gradient calculations reduce its overall speed-up factor to 10x-250x.

[6]Each NORB example is a binocular pair of 96x96 images. To reduce processing time, we downsampled each 96x96 image to 32x32 pixels. Hence, each simple unit sees 128 pixels from an 8x8 patch from each of the two binocular images. The input was whitened using ZCA (Zero-Phase Components Analysis).

[7]We used an SVM [22] as the linear classifier and determined $C$ by cross-validation over $\{10^{-4}, 10^{-3}, \ldots, 10^{4}\}$. Models were trained with various untied map sizes $k \in \{1, 2, 9, 16, 25\}$ and number of maps $l \in \{4, 6, 10, 16\}$. When $k = 1$, we were able to use an efficient convolutional implementation to scale up the number of maps in the models, allowing us to train additional models with $l \in \{22, 36, 48\}$.

[8]Each CIFAR-10 example is a 32x32 RGB image, also whitened using ZCA. Hence, each simple unit sees three patches from three channels of the color image input (RGB).

[9]These visualizations are available at `http://ai.stanford.edu/~quocle/`.

# References

[1] Y. LeCun, L. Bottou, Y. Bengio, and P. Haffner. Gradient based learning applied to document recognition. *Proceeding of the IEEE*, 1998.

[2] P. Simard, D. Steinkraus, and J. Platt. Best practices for convolutional neural networks applied to visual document analysis. In *ICDAR*, 2003.

[3] Y. LeCun, F.J. Huang, and L. Bottou. Learning methods for generic object recognition with invariance to pose and lighting. In *CVPR*, 2004.

[4] R. Collobert and J. Weston. A unified architecture for natural language processing: Deep neural networks with multitask learning. In *ICML*, 2008.

[5] Rajat Raina, Alexis Battle, Honglak Lee, Benjamin Packer, and Andrew Y. Ng. Self-taught learning: Transfer learning from unlabeled data. In *ICML*, 2007.

[6] G.E. Hinton, S. Osindero, and Y.W. Teh. A fast learning algorithm for deep belief nets. *Neural Computation*, 2006.

[7] D. Erhan, A. Courville, Y. Bengio, and P. Vincent. Why does unsupervised pre-training help deep learning? *Journal of Machine Learning Research*, 2010.

[8] A. Hyvarinen and P. Hoyer. Topographic independent component analysis as a model of V1 organization and receptive fields. *Neural Computation*, 2001.

[9] A. Hyvarinen, J. Hurri, and P. Hoyer. *Natural Image Statistics*. Springer, 2009.

[10] A. Krizhevsky. Learning multiple layers of features from tiny images. Technical report, U. Toronto, 2009.

[11] H. Lee, R. Grosse, R. Ranganath, and A.Y. Ng. Convolutional deep belief networks for scalable unsupervised learning of hierarchical representations. In *ICML*, 2009.

[12] M.A. Ranzato K. Jarrett, K. Kavukcuoglu and Y. LeCun. What is the best multi-stage architecture for object recognition? In *ICCV*, 2009.

[13] I. Goodfellow, Q.V. Le, A. Saxe, H. Lee, and A.Y. Ng. Measuring invariances in deep networks. In *NIPS*, 2010.

[14] B. Olshausen and D. Field. Emergence of simple-cell receptive field properties by learning a sparse code for natural images. *Nature*, 1996.

[15] A. Hyvarinen, J. Karhunen, and E. Oja. *Independent Component Analysis*. Wiley Interscience, 2001.

[16] A. Hyvarinen. Estimation of non-normalized statistical models using score matching. *JMLR*, 2005.

[17] K. Kavukcuoglu, M.A. Ranzato, R. Fergus, and Y. LeCun. Learning invariant features through topographic filter maps. In *CVPR*, 2009.

[18] Y. Bengio, P. Lamblin, D. Popovici, and H. Larochelle. Greedy layerwise training of deep networks. In *NIPS*, 2007.

[19] V. Nair and G. Hinton. 3D object recognition with deep belief nets. In *NIPS*, 2009.

[20] Y. Bengio and Y. LeCun. Scaling learning algorithms towards AI. In *Large-Scale Kernel Machines*, 2007.

[21] R. Salakhutdinov and H. Larochelle. Efficient learning of Deep Boltzmann Machines. In *AISTATS*, 2010.

[22] R.E. Fan, K.W. Chang, C.J. Hsieh, X.R. Wang, and C.J. Lin. LIBLINEAR: A library for large linear classification. *JMLR*, 9:1871–1874, 2008.

[23] G. Hinton and R. Salakhutdinov. Reducing the dimensionality of data with neural networks. *Science*, 2006.

[24] M. Ranzato and G. Hinton. Modeling pixel means and covariances using factorized third-order boltzmann machines. In *CVPR*, 2010.

[25] K. Yu and T. Zhang. Improved local coordinate coding using local tangents. In *ICML*, 2010.

[26] Y. Bengio. Learning deep architectures for AI. *Foundations and Trends in Machine Learning*, 2009.

[27] A. Saxe, M. Bhand, Z. Chen, P. W. Koh, B. Suresh, and A. Y. Ng. On random weights and unsupervised feature learning. In *Workshop: Deep Learning and Unsupervised Feature Learning (NIPS)*, 2010.

[28] D. Erhan, Y. Bengio, A. Courville, and P. Vincent. Visualizing higher-layer features of a deep network. Technical report, University of Montreal, 2009.

[29] P. Berkes and L. Wiskott. Slow feature analysis yields a rich repertoire of complex cell properties. *Journal of Vision*, 2005.

[30] R. Fergus A. Torralba and W. T. Freeman. 80 million tiny images: a large dataset for non-parametric object and scene recognition. In *IEEE Transactions on Pattern Analysis and Machine Intelligence*, 2008.

